# Constraint Classification for Multiclass Classification and Ranking

**Sariel Har-Peled**          **Dan Roth**          **Dav Zimak**

Department of Computer Science
University of Illinois
Urbana, IL 61801
{sariel,danr,davzimak}@uiuc.edu

## Abstract

The constraint classification framework captures many flavors of multiclass classification including winner-take-all multiclass classification, multilabel classification and ranking. We present a meta-algorithm for learning in this framework that learns via a single linear classifier in high dimension. We discuss distribution independent as well as margin-based generalization bounds and present empirical and theoretical evidence showing that constraint classification benefits over existing methods of multiclass classification.

## 1  Introduction

Multiclass classification is a central problem in machine learning, as applications that require a discrimination among several classes are ubiquitous. In machine learning, these include handwritten character recognition [LS97, LBD+89], part-of-speech tagging [Bri94, EZR01], speech recognition [Jel98] and text categorization [ADW94, DKR97].

While binary classification is well understood, relatively little is known about multiclass classification. Indeed, the most common approach to multiclass classification, the *one-versus-all* (OvA) approach, makes direct use of standard binary classifiers to encode and train the output labels. The OvA scheme assumes that for each class there exists a single (simple) separator between that class and all the other classes. Another common approach, *all-versus-all* (AvA) [HT98], is a more expressive alternative which assumes the existence of a separator between any two classes.

OvA classifiers are usually implemented using a winner-take-all (WTA) strategy that associates a real-valued function with each class in order to determine class membership. Specifically, an example belongs to the class which assigns it the highest value (i.e., the "winner") among all classes. While it is known that WTA is an expressive classifier [Maa00], it has limited expressivity when trained using the OvA assumption since OvA assumes that each class can be easily separated from the rest. In addition, little is known about the generalization properties or convergence of the algorithms used.

This work is motivated by several successful practical approaches, such as multiclass support vector machines (SVMs) and the sparse network of winnows (SNoW) architecture that

rely on the WTA strategy over linear functions. Our aim is to improve the understanding of such classifier systems and to develop more theoretically justifiable algorithms that realize the full potential of WTA.

An alternative interpretation of WTA is that every example provides an ordering of the classes (sorted in descending order by the assigned values), where the "winner" is the first class in this ordering. It is thus natural to specify the ordering of the classes for an example *directly*, instead of implicitly through WTA.

In Section 2, we introduce *constraint classification*, where each example is labeled with a set of constraints relating multiple classes. Each such constraint specifies the relative order of two classes for this example. The goal is to learn a classifier consistent with these constraints. Learning is made possible by a simple transformation mapping each example into a set of examples (one for each constraint) and the application of any *binary* classifier on the mapped examples. In Section 3, we present a new algorithm for constraint classification that takes on the properties of the binary classification algorithm used. Therefore, using the Perceptron algorithm, it is able to learn a consistent classifier if one exists, using the winnow algorithm it can learn attribute efficiently, and using the SVM, it provides a simple implementation of multiclass SVM. The algorithm can be implemented with a subtle change to the standard (via OvA) approach to training a network of linear threshold gates. In Section 4, we discuss both VC-dimension and margin-based generalization bounds presented a companion paper[HPRZ02]. Our generalization bounds apply to WTA classifiers over linear functions, for which VC-style bounds were not known.

In addition to multiclass classification, constraint classification generalizes multilabel classification, ranking on labels, and of course, binary classification. As a result, our algorithm provides new insight into these problems, as well as new, powerful tools for solving them. For example, in Section , we show that the commonly used OvA assumption can cause learning to fail, even when a consistent classifier exists. Section 5 provides empirical evidence that the constraint classification outperforms the OvA approach.

## 2   Constraint Classification

Learning problems often assume that examples, $(x, y) \in \mathcal{X} \times \mathcal{Y}$, are drawn *i.i.d.* from fixed probability distribution, $\mathcal{D}_{\mathcal{X} \times \mathcal{Y}}$, over $\mathcal{X} \times \mathcal{Y}$. $\mathcal{X}$ is referred to as the instance space and $\mathcal{Y}$ is referred to as the output space (*label set*).

**Definition 2.1 (Learning)**  Given $m$ examples, $S = ((x_1, y_1), \ldots, (x_m, y_m))$, drawn *i.i.d.* from $\mathcal{D}_{\mathcal{X} \times \mathcal{Y}}$, a hypothesis class $\mathcal{H}$ and an error function $\mathcal{E} : \mathcal{X} \times \mathcal{Y} \times \mathcal{H} \to \{0, 1\}$, a learning algorithm $\mathcal{L}(S, \mathcal{H})$ attempts to output a function $h \in \mathcal{H}$, where $h : \mathcal{X} \to \mathcal{Y}$, that minimizes the expected error on a randomly drawn example.

**Definition 2.2 (Permutations)**  Denote the set of full orders over $\{1, \ldots, k\}$ as $S^k$, consisting of all permutations of $\{1, \ldots, k\}$. Similarly, $\bar{S}^k$ denotes the set of all partial orders over $\{1, \ldots, k\}$. A partial order, $c \in \bar{S}^k$, defines a binary relation, $\prec_c$ and can be represented by set of pairs on which $\prec_c$ holds, $c = \{(i, j) | i \prec_c j\}$. In addition, for any set of pairs $c = \{(i_1, j_1), \ldots, (i_n, j_n)\}$, we refer to $c$ both as a set of pairs and as the partial order produced by the transitive closure of $c$ with respect to $\prec_c$. Given two partial orders $a, b \in \bar{S}^k$, $a$ is *consistent* with $b$ (denoted $a \sqsubseteq b$) if for every $(i, j) \in \{1, \ldots, k\}^2$, $i \prec_b j$ holds whenever $i \prec_a j$. If $c \in S^k$ is a full order, then it can be represented by a list of $k$ integers where $i \prec_c j$ if $i$ precedes $j$ in the list. The size of a partial order, $|c|$ is the number of pairs specified in $c$.

**Definition 2.3 (Constraint Classification)**  Constraint classification is a learning problem where each example $(x, y) \in \mathcal{X} \times \bar{S}^k$ is labeled according to a partial order $y \in \bar{S}^k$. A constraint classifier, $h : \mathcal{X} \to \bar{S}^k$, is consistent with example $(x, y)$ if $y$ is consistent with $h(x)$ ($y \sqsubseteq h(x)$). When $|y| \leq c$, we call it *c-constraint classification*.

| Problem | Internal Representation | Output Space ($\mathcal{Y}$) | Hypothesis | Size of Mapping |
|---|---|---|---|---|
| binary | $w \in \mathbb{R}^d$ | $\{-1, 1\}$ | $\operatorname{sign} w \cdot x$ | 1 |
| multiclass | $(w_1, \ldots, w_k) \in \mathbb{R}^{kd}$ | $\{1, \ldots, k\}$ | $\operatorname{argmax}_{\{1,\ldots,k\}} w_i \cdot x$ | $k-1$ |
| $l$-multilabel | $(w_1, \ldots, w_k) \in \mathbb{R}^{kd}$ | $\{1, \ldots, k\}^l$ | $\operatorname{argmax}^l_{\{1,\ldots,k\}} w_i \cdot x$ | $l(k-l)$ |
| ranking | $(w_1, \ldots, w_k) \in \mathbb{R}^{kd}$ | $S^k$ | $\operatorname{argsort}_{\{1,\ldots,k\}} w_i \cdot x$ | $k-1$ |
| constraint* | $(w_1, \ldots, w_k) \in \mathbb{R}^{kd}$ | $\bar{S}^k$ | $\operatorname{argsort}_{\{1,\ldots,k\}} w_i \cdot x$ | – |
| $c$-constraint* | $(w_1, \ldots, w_k) \in \mathbb{R}^{kd}$ | $\bar{S}^k_c$ | $\operatorname{argsort}_{\{1,\ldots,k\}} w_i \cdot x$ | $c$ |

Table 1: Definitions for various learning problems (notice that the hypothesis for constraint classification is always a full order) and the size of the resultant mapping to $c$-constraint classification. $\operatorname{argmax}^l$ is a variant of $\operatorname{argmax}$ that returns the $l$ maximal indices with respect to $w_i \cdot x$. argsort is a linear sorting function (see Definition 2.6).

**Definition 2.4 (Error Indicator Function)** For any $(x, y) \in \mathcal{X} \times \bar{S}^k$, and hypothesis $h : \mathcal{X} \to \bar{S}^k$, the *indicator function* $\mathcal{E}(x, y, h)$ indicates an error on example $x$, $\mathcal{E}(x, y, h) = 1$ if $y \not\sqsubseteq h(x)$, and 0 otherwise.

For example, if $k = 4$ and example $(x, y) = (x, \{(2, 3), (2, 4)\})$, $h_1(x) = (2, 3, 1, 4)$, and $h_2(x) = (4, 2, 3, 1)$, then $h_1$ is correct since 2 precedes 3 and 2 precedes 4 in the full order $(2, 3, 1, 4)$ whereas $h_2$ is incorrect since 4 precedes 2 in $(4, 2, 3, 1)$.

**Definition 2.5 (Error)** Given an example $(x, y)$ drawn $i.i.d.$ from $\mathcal{D}_{\mathcal{X} \times \mathcal{Y}}$, the *true error* of $h \in \mathcal{H}$, where $h : \mathcal{X} \to \mathcal{Y}'$ is defined to be $err(h) = \operatorname{Pr}_{\mathcal{D}}[\mathcal{E}(x, y, h)]$. Given $S = ((x_1, y_1), \ldots, (x_m, y_m))$, the *empirical error* of $h \in \mathcal{H}$ with respect to $S$ is defined to be $err(S, h) = \frac{1}{|S|} |\sum_{(x,y) \in S} \mathcal{E}(x, y, h)|$.

In this paper, we consider constraint classification problems where hypotheses are functions from $\mathbb{R}^d$ to $S^k$ that output a permutation of $\{1, \ldots, k\}$.

**Definition 2.6 (Linear Sorting Function)** Let $w = (w_1, \ldots, w_k)$ be a set of $k$ vectors, where $(w_1, \ldots, w_k) \in \mathbb{R}^d$. Given $x \in \mathbb{R}^d$, a *linear sorting classifier* is a function $h : \mathbb{R}^d \to S^k$ computed in the following way:

$$h(x) = \operatorname*{argsort}_{i=1 \ldots k} w_i \cdot x,$$

where argsort returns a permutation of $\{1, \ldots, k\}$ where $i$ precedes $j$ if $w_i \cdot x > w_j \cdot x$. In the case that $w_i \cdot x = w_j \cdot x$, $i$ precedes $j$ if $i < j$.

Constraint classification can model many well-studied learning problems including multiclass classification, ranking and multilabel classification. Table 1 shows a few interesting classification problems expressible as constraint classification. It is easy to show:

**Lemma 2.7 (Problem mappings)** *All of the learning problems in Table 1 can be expressed as constraint classification problems.*

Consider a 4-class multiclass example, $(x, 3)$. It is transformed into the 3-constraint example, $(x, \{(3, 1), (3, 2), (3, 4)\})$. If we find a constraint classifier that correctly labels $x$ according to the given constraints where $w_3 \cdot x > w_1 \cdot x$, $w_3 \cdot x > w_2 \cdot x$, and $w_3 \cdot x > w_4 \cdot x$, then $3 = \operatorname{argmax}_{1,2,3,4} w_i \cdot x$. If instead we are given a ranking example $(x, \{(3, 2, 1, 4)\})$, it can be transformed into $(x, \{(3, 2), (2, 1), (1, 4)\})$.

## 3 Learning

In this section, $k$-class constraint classification is transformed into binary classification in higher dimension. Each example $(x, y) \in \mathbb{R}^d \times \bar{S}^k$ becomes a set of examples in $\mathbb{R}^{kd} \times$

$\{-1, 1\}$ with each constraint $(i, j)$ contributing a single 'positive' and a single 'negative' example. Then, a separating hyperplane for the expanded example set (in $\mathbb{R}^{kd}$) can be viewed as a linear sorting function over $k$ linear functions, each in $d$ dimensional space.

## 3.1 Kesler's Construction

Kesler's construction for multiclass classification was first introduced by Nilsson in 1965[Nil65, 75–77] and can also be found more recently[DH73]. This subsection extends the Kesler construction for constraint classification.

**Definition 3.1 (Chunk)** A vector $\mathbf{v} = (v_1, \ldots, v_{kd}) \in \mathbb{R}^{kd} = \mathbb{R}^d \times \cdots \times \mathbb{R}^d$, is broken into $k$ *chunks* $(\mathbf{v}_1, \ldots, \mathbf{v}_k)$ where the $i$-th chunk, $\mathbf{v}_i = (v_{(i-1)*d+1}, \ldots, v_{i*d})$.

**Definition 3.2 (Expansion)** Let $\text{Vec}(x, i)$ be a vector $x \in \mathbb{R}^d$ embedded in $kd$ dimensions, by writing the coordinates of $x$ in the $i$-th chunk of a vector in $\mathbb{R}^{k(d+1)}$. Denote by $\mathbf{0}^l$ the zero vector of length $l$. Then $\text{Vec}(x, i)$ can be written as the concatenation of three vectors, $\text{Vec}(x, i) = (\mathbf{0}^{(i-1)*d}, x, \mathbf{0}^{(k-i)*d}) \in \mathbb{R}^{kd}$. Finally, $\text{Vec}(x, i, j) = \text{Vec}(x, i) - \text{Vec}(x, j)$, is the embedding of $x$ in the $i$-th chunk and $-x$ in the $j$-th chunk of a vector in $\mathbb{R}^{kd}$.

**Definition 3.3 (Expanded Example Sets)** Given an example $(x, y)$, where $x \in \mathbb{R}^d$ and $y \in \bar{S}^k$, we define the *expansion* of $(x, y)$ into a set of examples as follows,

$$\mathbf{P}_+(x, y) = \left\{ (\text{Vec}(x, i, j), 1) \;\middle|\; (i, j) \in y \right\} \subseteq \mathbb{R}^{kd} \times \{1\},$$

A set of negative examples is defined as the reflection of each expanded example through the origin, specifically

$$\mathbf{P}_-(x, y) = \left\{ (-\mathbf{x}, -1) \;\middle|\; (\mathbf{x}, 1) \in \mathbf{P}_+(x, y) \right\} \subseteq \mathbb{R}^{kd} \times \{-1\},$$

and the set of both positive and negative examples is denoted by $\mathbf{P}(x, y) = \mathbf{P}_+(x, y) \cup \mathbf{P}_-(x, y)$. The expansion of a set of examples, $S$, is defined as the union of all of the expanded examples in the set,

$$\mathbf{P}(S) = \bigcup_{(x, y) \in S} \mathbf{P}(x, y) \subseteq \mathbb{R}^{kd} \times \{-1, 1\}.$$

Note that the original Kesler construction produces only $\mathbf{P}_+$. We also create $\mathbf{P}_-$ to simplify the analysis and to maintain consistency when learning non-linear functions (such as SVM).

## 3.2 Algorithm

Figure 1 (a) shows a meta-learning algorithm for constraint classification that finds a linear sorting function by using any algorithm for learning a binary classifier. Given a set of examples $S \subseteq \mathbb{R}^d \times \bar{S}^k$, the algorithm simply finds a separating hyperplane $h(\mathbf{x}) = \mathbf{v} \cdot \mathbf{x}$ for $\mathbf{P}(S) \subseteq \mathbb{R}^{kd} \times \{-1, 1\}$. Suppose $h$ correctly classifies $(\mathbf{x}, 1) = (\text{Vec}(x, i, j), 1) \in \mathbf{P}(S)$, then $\mathbf{x} \cdot \mathbf{v} = x \cdot \mathbf{v}_i - x \cdot \mathbf{v}_j > 0$, and the constraint $(i, j)$ on $x$ (dictating that $x \cdot \mathbf{v}_i > x \cdot \mathbf{v}_j$) is consistent with $h(\mathbf{x})$. Therefore, if $h(\mathbf{x})$ correctly classifies all $x \in \mathbf{P}(S)$, then $\text{argsort}_{1,\ldots,k} \mathbf{v}_i \cdot x$ is a consistent linear sorting function.

This framework is significant to multiclass classification in many ways. First, the hypothesis learned above is more expressive than when the OvA assumption is used. Second, it is easy to verify that other algorithmic-specific properties are maintained by the above transformation. For example, attribute efficiency is preserved when using the winnow algorithm. Finally, the multiclass support vector machine can be implemented by learning a hyperplane to separate $\mathbf{P}(S)$ with maximal margin.

## 3.3 Comparison to "One-Versus-All"

A common approach to multiclass classification ($\mathcal{Y} = \{1, \ldots, k\}$) is to make the *one-versus-all (OvA)* assumption, namely, that each class can be separated from the rest using

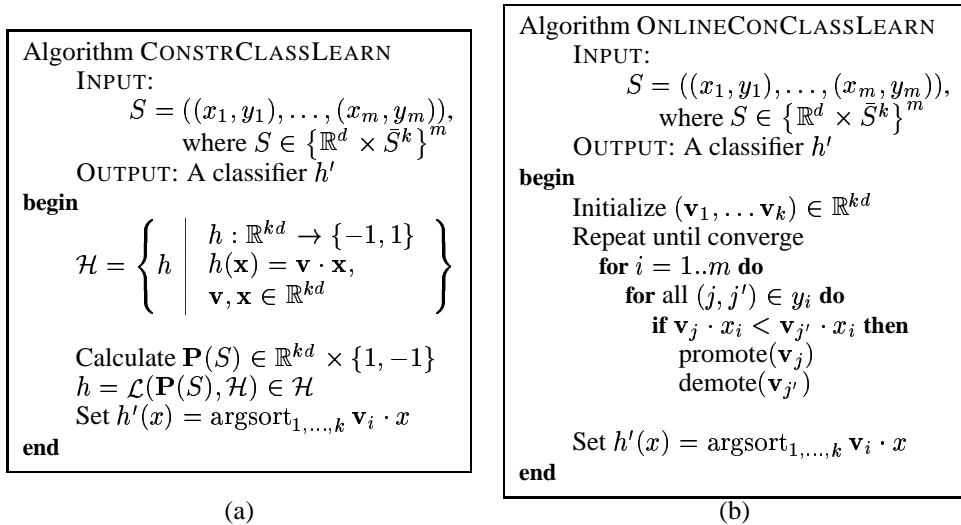

Algorithm CONSTRCLASSLEARN
  INPUT:
    $S = ((x_1, y_1), \ldots, (x_m, y_m))$,
      where $S \in \{\mathbb{R}^d \times \bar{S}^k\}^m$
  OUTPUT: A classifier $h'$
**begin**

$$\mathcal{H} = \left\{ h \;\middle|\; \begin{array}{l} h : \mathbb{R}^{kd} \to \{-1, 1\} \\ h(\mathbf{x}) = \mathbf{v} \cdot \mathbf{x}, \\ \mathbf{v}, \mathbf{x} \in \mathbb{R}^{kd} \end{array} \right\}$$

  Calculate $\mathbf{P}(S) \in \mathbb{R}^{kd} \times \{1, -1\}$
  $h = \mathcal{L}(\mathbf{P}(S), \mathcal{H}) \in \mathcal{H}$
  Set $h'(x) = \text{argsort}_{1, \ldots, k} \mathbf{v}_i \cdot x$
**end**

(a)

Algorithm ONLINECONCLASSLEARN
  INPUT:
    $S = ((x_1, y_1), \ldots, (x_m, y_m))$,
      where $S \in \{\mathbb{R}^d \times \bar{S}^k\}^m$
  OUTPUT: A classifier $h'$
**begin**
  Initialize $(\mathbf{v}_1, \ldots \mathbf{v}_k) \in \mathbb{R}^{kd}$
  Repeat until converge
    **for** $i = 1..m$ **do**
      **for** all $(j, j') \in y_i$ **do**
        **if** $\mathbf{v}_j \cdot x_i < \mathbf{v}_{j'} \cdot x_i$ **then**
          promote($\mathbf{v}_j$)
          demote($\mathbf{v}_{j'}$)

  Set $h'(x) = \text{argsort}_{1, \ldots, k} \mathbf{v}_i \cdot x$
**end**

(b)

Figure 1: (a) Meta-learning algorithm for constraint classification with linear sorting functions (see Definition 2.6). $\mathcal{L}(\cdot, \cdot)$ is any binary learning algorithm returning a separating hyperplane. (b) Online meta-algorithm for constraint classification with linear sorting functions (see Definition 2.6). The particular online algorithm used determines how $(\mathbf{v}_1, \ldots, \mathbf{v}_k)$ is initialized and the promotion and demotion strategies.

a binary classification algorithm. Learning proceeds by learning $k$ independent binary classifiers, one corresponding to each class, where example $(x, y)$ is considered positive for classifier $y$ and negative for all others.

It is easy to construct an example where the OvA assumption causes the learning to fail even when there exists a consistent linear sorting function. (see Figure 2) Notice, since the existence of a consistent linear sorting function (w.r.t. $S$) implies the existence of a separating hyperplane (w.r.t. $\mathbf{P}(S)$), any learning algorithm guaranteed to separate two separable point sets (e.g. the Perceptron algorithm) is guaranteed to find a consistent linear sorting function. In Section 5, we use the perceptron algorithm to find a consistent classifier for an extension of the example in Figure 2 to $\mathbb{R}^{100}$ when OvA fails.

### 3.4   Comparison to Newtorks of Linear Threshold Gates (Perceptron)

It is possible to implement the algorithm in Section  using a network of linear classifiers such as multi-output Perceptron [AB99], SNoW [CCRR99, Rot98], and multiclass SVM [CS00, WW99]. Such a network has $x \in \mathbb{R}^d$ as input and $k$ outputs, each represented by a weight vector, $w_i \in \mathbb{R}^d$, where the $i$-th output computes $w_i \cdot x$ (see Figure 1 (b)).

Typically, a label is mapped, via fixed transformation, into a $k$-dimensional output vector, and each output is trained separately, as in the OvA case. Alternately, if the online perceptron algorithm is plugged into the meta-algorithm in Section , then updates are performed according to a dynamic transformation. Specifically, given $(x, y)$, for every constraint $(i, j) \in y$, if $w_i \cdot x < w_j \cdot x$, $w_i$ is 'promoted' and $w_j$ is 'demoted'. Using a network in this results in an ultraconservative online algorithm for multiclass classification [CS01]. This subtle change enables the commonly used network of linear threshold gates to learn every hypothesis it is capable of representing.

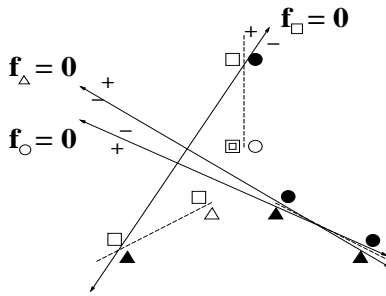

Figure 2: A 3-class classification example in $\mathbb{R}^2$ showing that one-versus-all (OvA) does not converge to a consistent hypothesis. Three classes (squares, triangles, and circles) should be separated from the rest. Solid points act as 10 points in their respective classes. The OvA assumption will attempt to separate the circles from squares and triangles with a single separating hyperplane, as well as the other 2 combinations. Because the solid points are weighted, all OvA classifiers are required to classify them correctly or suffer 10 mistakes, thus restricting what the final hypotheses will be. As a result, the OvA assumption will misclassify point outlined with a double square since the square classifier predicts "not square" and the circle classifier predicts "circle". One can verify that there exists a WTA classifier for this example.

| Dataset | Features | Classes | Training Examples | Testing Examples |
|---|---|---|---|---|
| glass | 9 | 6 | 214 | – |
| vowel | 10 | 11 | 528 | 462 |
| soybean | 35 | 19 | 307 | 376 |
| audiology | 69 | 24 | 200 | 26 |
| ISOLET | 617 | 26 | 6238 | 1559 |
| letter | 16 | 26 | 16000 | 4000 |
| Synthetic* | 100 | 3 | 50000 | 50000 |

Table 2: Summary of problems from the UCI repository. The synthetic data is sampled from a random linear sorting function (see Section 5).

## 4    Generalization Bounds

A PAC-style analysis of multiclass functions that uses an extended notion of VC-dimension for multiclass case [BCHL95] provides poor bounds on generalization for WTA, and the current best bounds rely on a generalized notion of margin [ASS00]. In this section, we prove tighter bounds using the new framework.

We seek generalization bounds for learning with $\mathcal{H}$, the class of linear sorting functions (Definition 2.6). Although both VC-dimension-based (based on growth function) and margin-based bounds for the class of hyperplanes in $\mathbb{R}^{kd}$ are known [Vap98, AB99], they cannot directly be applied since $\mathbf{P}(S)$ produces points that are random, but not *independently* drawn. It turns out that bounds can be derived indirectly by using known bounds for constraint classification. Due to space considerations see[HPRZ02], where natural extensions to the growth function and margin are used to develop generalization bounds.

## 5    Experiments

As in previous multiclass classification work [DB95, ASS00], we tested our algorithm on a suite of problems from the Irvine Repository of machine learning [BM98] (see Table 2). In addition, we created a simple experiment using synthetic data. The data was generated according to a WTA function over 3 randomly generated linear functions in $\mathbb{R}^1 00$, each with weight vectors inside the unit ball. Then, 50K training and 50K testing examples were

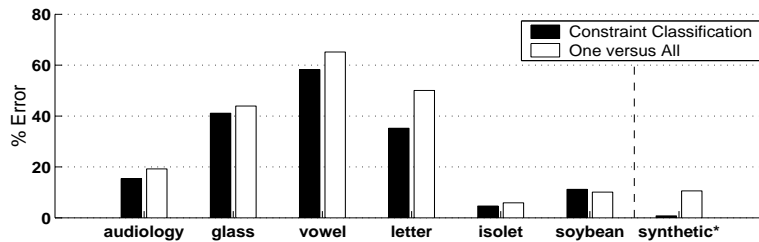

Figure 3: Comparison of constraint classification meta-algorithm using the Perceptron algorithm to multi-output Perceptron using the OvA assumption. All of the results for the constraint classification algorithm are competitive with the known. The synthetic data would converge to 0 error using constraint classification but would not converge using the OvA approach.

randomly sampled within a ball of radius 2 around the origin and labeled with the linear function that produced the highest value.

A comparison is made between the OvA approach (Section ) and the constraint classification approach. Both were implemented on the same network of multi-output Perceptron network with $k(d+1)$ weights (with one threshold per class). Constraint classification used the modified update rule discussed in Section . Each update was performed as follows: $w_{t+1} = w_t + x$ for promotion and $w_{t+2} = w_t - x$ for demotion. The networks were initialized with weights all 0.

For each multiclass example $(x, y_m) \in \mathbb{R}^d \times \{1, \ldots, k\}$, a constraint classification example $(x, y_c) \in \mathbb{R}^d \times \bar{S}^k$ was created, where $y_c = \left\{ (y_m, i) \mid j = \{1, \ldots, k\} \setminus y_m \right\}$. Notice error (Definition 2.4) of $(x, y_c)$ corresponds to the traditional error for multiclass classification.

Figure 3 shows that constraint classification outperforms the multioutput Perceptron when using the OvA assumption.

# 6 Discussion

We think constraint classification provides two significant contributions to multiclass classification. Firstly, it provides a conceptual generalization that encompasses multiclass classification, multilabel classification, and label ranking problems in addition to problems with more complex relationships between labels. Secondly, it reminds the community that the Kesler construction can be used to extend *any* learning algorithm for binary classification to the multiclass (or constraint) setting.

Section 5 showed that the constraint approach to learning is advantageous over the one-versus-all approach on both real-world and synthetic data sets. However, preliminary experiments using various natural language data sets, such as part-of-speech tagging, do not yield any significant difference between the two approaches. We used a common transformation [EZR01] to convert raw data to approximately three million examples in one hundred thousand dimensional boolean feature space. There were about 50 different part-of-speech tags. Because the constraint approach is more expressive than the one-versus-all approach, and because both approaches use the same hypothesis space ($k$ linear functions), we expected the constraint approach to achieve higher accuracy. Is it possible that a difference would emerge if more data were used? We find it unlikely since both methods use identical representations. Perhaps, it is instead a result of the fact that we are working in very high dimensional space. Again, we think this is not the case, since it seems that "most" random winner-take-all problems (as with the synthetic data) would cause the one-versus-all assumption to fail.

Rather, we conjecture that for some reason, natural language problems (along with the

transformation) are suited to the one-versus-all approach and do not require a more complex hypothesis. Why, and how, this is so is a direction for future speculation and research.

# 7   Conclusions

The view of multiclass classification presented here simplifies the implementation, analysis, and understanding of many preexisting approaches. Multiclass support vector machines, ultraconservative online algorithms, and traditional one-versus-all approaches can be cast in this framework. It would be interesting to see if it could be combined with the error-correcting output coding method in [DB95] that provides another way to extend the OvA approach. Furthermore, this view allows for a very natural extension of multiclass classification to constraint classification – capturing within it complex learning tasks such as multilabel classification and ranking. Because constraint classification is a very intuitive approach and its implementation can be carried out by any discriminant technique, and not only by optimization techniques, we think it will have useful real-world applications.

# References

[AB99]    M. Anthony and P. Bartlett. *Neural Network Learning: Theoretical Foundations*. Cambridge University Press, Cambridge, England, 1999.

[ADW94]   C. Apte, F. Damerau, and S. M. Weiss. Automated learning of decision rules for text categorization. *Information Systems*, 12(3):233–251, 1994.

[ASS00]   E. Allwein, R.E. Schapire, and Y. Singer. Reducing multiclass to binary: A unifying approach for margin classifiers. In *Proc. 17th International Conf. on Machine Learning*, pages 9–16. Morgan Kaufmann, San Francisco, CA, 2000.

[BCHL95]  S. Ben-David, N. Cesa-Bianchi, D. Haussler, and P. Long. Characterizations of learnability for classes of $0, \ldots, n$-valued functions. *J. Comput. Sys. Sci.*, 50(1):74–86, 1995.

[BM98]    C.L. Blake and C.J. Merz. UCI repository of machine learning databases, 1998.

[Bri94]   E. Brill. Some advances in transformation-based part of speech tagging. In *AAAI, Vol. 1*, pages 722–727, 1994.

[CCRR99]  A. Carlson, C. Cumby, J. Rosen, and D. Roth. The SNoW learning architecture. Technical Report UIUCDCS-R-99-2101, UIUC Computer Science Department, May 1999.

[CS00]    K. Crammer and Y. Singer. On the learnability and design of output codes for multiclass problems. In *Computational Learing Theory*, pages 35–46, 2000.

[CS01]    K. Crammer and Y. Singer. Ultraconservative online algorithms for multiclass problems. In *COLT/EuroCOLT*, pages 99–115, 2001.

[DB95]    T. Dietterich and G. Bakiri. Solving multiclass learning problems via error-correcting output codes. *Journal of Artificial Intelligence Research*, 2:263–286, 1995.

[DH73]    R. Duda and P. Hart. *Pattern Classification and Scene Analysis*. Wiley, New York, 1973.

[DKR97]   I. Dagan, Y. Karov, and D. Roth. Mistake-driven learning in text categorization. In *EMNLP-97, The Second Conference on Empirical Methods in Natural Language Processing*, pages 55–63, 1997.

[EZR01]   Y. Even-Zohar and D. Roth. A sequential model for multi class classification. In *EMNLP-2001, the SIGDAT Conference on Empirical Methods in Natural Language Processing*, pages 10–19, 2001.

[HPRZ02]  S. Har-Peled, D. Roth, and D. Zimak. Constraint classification: A new approach to multiclass classification. In *Proc. 13th International Conf. of Algorithmic Learning Theory*, pages 365–397, 2002.

[HT98]    T. Hastie and R. Tibshirani. Classification by pairwise coupling. In *NIPS-10, The 1997 Conference on Advances in Neural Information Processing Systems*, pages 507–513. MIT Press, 1998.

[Jel98]   F. Jelinek. *Statistical Methods for Speech Recognition*. The MIT Press, Cambridge, Massachusetts, 1998.

[LBD+89]  Y. Le Cun, B. Boser, J. Denker, D. Hendersen, R. Howard, W. Hubbard, and L. Jackel. Backpropagation applied to handwritten zip code recognition. *Neural Computation*, 1:pp 541, 1989.

[LS97]    D. Lee and H. Seung. Unsupervised learning by convex and conic coding. In Michael C. Mozer, Michael I. Jordan, and Thomas Petsche, editors, *Advances in Neural Information Processing Systems*, volume 9, page 515. The MIT Press, 1997.

[Maa00]   W. Maass. On the computational power of winner-take-all. *Neural Computation*, 12(11):2519–2536, 2000.

[Nil65]   Nils J. Nilsson. *Learning Machines: Foundations of trainable pattern-classifying systems*. McGraw-Hill, New York, NY, 1965.

[Rot98]   D. Roth. Learning to resolve natural language ambiguities: A unified approach. In *Proc. of AAAI*, pages 806–813, 1998.

[Vap98]   V. Vapnik. *Statistical Learning Theory*. Wiley, 605 Third Avenue, New York, New York, 10158-0012, 1998.

[WW99]    J. Weston and C. Watkins. Support vector machines for multiclass pattern recognition. In *Proceedings of the Seventh European Symposium On Artificial Neural Networks*, 4 1999.
